# Trait selection for assessing beef meat quality using non-linear SVM

**J.J. del Coz, G. F. Bayón, J. Díez,**
**O. Luaces, A. Bahamonde**
Artificial Intelligence Center
University of Oviedo at Gijón
juanjo@aic.uniovi.es

**Carlos Sañudo**
Facultad de Veterinaria
University of Zaragoza
csanudo@posta.unizar.es

## Abstract

In this paper we show that it is possible to model sensory impressions of consumers about beef meat. This is not a straightforward task; the reason is that when we are aiming to induce a function that maps object descriptions into ratings, we must consider that consumers' ratings are just a way to express their preferences about the products presented in the same testing session. Therefore, we had to use a special purpose SVM polynomial kernel. The training data set used collects the ratings of panels of experts and consumers; the meat was provided by 103 bovines of 7 Spanish breeds with different carcass weights and aging periods. Additionally, to gain insight into consumer preferences, we used feature subset selection tools. The result is that aging is the most important trait for improving consumers' appreciation of beef meat.

## 1   Introduction

The quality of beef meat is appreciated through sensory impressions, and therefore its assessment is very subjective. However, it is known that there are objective traits very important for the final properties of beef meat; this includes the breed and feeding of animals, weight of carcasses, and aging of meat after slaughter. To discover the influence of these and other attributes, we have applied Machine Learning tools to the results of an experience reported in [8]. In the experience, 103 bovines of 7 Spanish breeds were slaughtered to obtain two kinds of carcasses, light and standard [5]; the meat was prepared with 3 aging periods, 1, 7, and 21 days. Finally, the meat was consumed by a group, called *panel*, of 11 experts, and assessed by a panel of untrained consumers.

The conceptual framework used for the study reported in this paper was the analysis of sensory data. In general, this kind of analysis is used for food industries in order to adapt their productive processes to improve the acceptability of their specialties. They need to discover the relationship between descriptions of their products and consumers' sensory degree of satisfaction. An excellent survey of the use of sensory data analysis in the food industry can be found in [15, 2]; for a Machine Learning perspective, see [3, 9, 6].

The role played by each panel, experts and consumers, is very clear. So, the experts' panel is made up of a usually small group of trained people who rate several traits of products such

as fibrosis, flavor, odor, etc... The most essential property of expert panelists, in addition to their discriminatory capacity, is their own coherence, but not necessarily the uniformity of the group. Experts' panel can be viewed as a bundle of sophisticated sensors whose ratings are used to describe each product, in addition to other objective traits. On the other hand, the group of untrained consumers ($C$) are asked to rate their degree of acceptance or satisfaction about the tested products on a given scale. Usually, this panel is organized in a set of *testing sessions*, where a group of potential consumers assess some instances from a sample $E$ of the tested product. Frequently, each consumer only participates in a small number (sometimes only one) of testing sessions, usually in the same day.

In general, the success of sensory analysis relies on the capability to identify, with a precise description, a kind of product that should be reproducible as many times as we need to be tested for as many consumers as possible. Therefore, the study of beef meat sensory quality is very difficult. The main reason is that there are important individual differences in each piece of meat, and the repeatability of tests can be only partially ensured. Notice that from each animal there are only a limited amount of similar pieces of meat, and thus we can only provide pieces of a given breed, weight, and aging period. Additionally, it is worthy noting that the cost of acquisition of this kind of sensory data is very high.

The paper is organized as follows: in the next section we present an approach to deal with testing sessions explicitly. The overall idea is to look for a *preference* or *ranking function* able to reproduce the implicit ordering of products given by consumers instead of trying to predict the exact value of consumer ratings; such function must return higher values to those products with higher ratings. In Section 3 we show how some state of the art FSS methods designed for SVM (Support Vector Machines) with non-linear kernels can be adapted to preference learning. Finally, at the end of the paper, we return to the data set of beef meat to show how it is possible to explain consumer behavior, and to interpret the relevance of meat traits in this context.

## 2   Learning from sensory data

A straightforward approach to handle sensory data can be based on regression, where sensory descriptions of each object $x \in E$ are endowed with the degree of satisfaction $r(x)$ for each consumer (or the average of a group of consumers). However, this approach does not faithfully captures people's preferences [7, 6]: consumers' ratings actually express a relative ordering, so there is a kind of *batch effect* that often biases their ratings. Thus, a product could obtain a higher (lower) rating depending on if it is assessed together with worse (better) products. Therefore, information about batches tested by consumers in each rating session is a very important issue. On the other hand, more traditional approaches, such as testing some statistical hypotheses [16, 15, 2] require all available food products in sample $E$ to be assessed by the set of consumers $C$, a requisite very difficult to fulfill.

In this paper we use an approach to sensory data analysis based on learning *consumers' preferences*, see [11, 14, 1], where training examples are represented by *preference judgments*, i.e. pairs of vectors $(v, u)$ indicating that, for someone, object $v$ is preferable to object $u$. We will show that this approach can induce more useful knowledge than other approaches, like regression based methods. The main reason is due to the fact that preference judgments sets can represent more relevant information to discover consumers' preferences.

### 2.1   A formal framework to learn consumer preferences

In order to learn our preference problems, we will try to find a real *ranking function f* that maximizes the probability of having $f(v) > f(u)$ whenever $v$ is preferable to $u$ [11, 14, 1].

Our input data is made up of a set of ratings $(r_i(\boldsymbol{x}) : \boldsymbol{x} \in E_i)$ for $i \in C$. To avoid the *batch effect*, we will create a preference judgment set $PJ = \{\boldsymbol{v}_j > \boldsymbol{u}_j : j = 1, \ldots, n\}$ suitable for our needs just considering all pairs $(\boldsymbol{v}, \boldsymbol{u})$ such that objects $\boldsymbol{v}$ and $\boldsymbol{u}$ were presented in the same session to a given consumer $i$, and $r_i(\boldsymbol{v}) > r_i(\boldsymbol{u})$.

Thus, following the approach introduced in [11], we look for a function $F : \mathbb{R}^d \times \mathbb{R}^d \to \mathbb{R}$ such that

$$\forall \boldsymbol{x}, \boldsymbol{y} \in \mathbb{R}^d, F(\boldsymbol{x}, \boldsymbol{y}) > 0 \Leftrightarrow F(\boldsymbol{x}, \boldsymbol{0}) > F(\boldsymbol{y}, \boldsymbol{0}). \tag{1}$$

Then, the ranking function $f : \mathbb{R}^d \to \mathbb{R}$ can be simply defined by $f(\boldsymbol{x}) = F(\boldsymbol{x}, \boldsymbol{0})$.

As we have already constructed a set of preference judgments $PJ$, we can specify $F$ by means of the restrictions

$$F(\boldsymbol{v}_j, \boldsymbol{u}_j) > 0 \text{ and } F(\boldsymbol{u}_j, \boldsymbol{v}_j) < 0, \quad \forall j = 1, \ldots, n. \tag{2}$$

Therefore, we have a binary classification problem that can be solved using SVM. We follow the same steps as Herbrich *et al.* in [11], and define a kernel $\mathcal{K}$ as follows

$$\mathcal{K}(\boldsymbol{x}_1, \boldsymbol{x}_2, \boldsymbol{x}_3, \boldsymbol{x}_4) = k(\boldsymbol{x}_1, \boldsymbol{x}_3) - k(\boldsymbol{x}_1, \boldsymbol{x}_4) - k(\boldsymbol{x}_2, \boldsymbol{x}_3) + k(\boldsymbol{x}_2, \boldsymbol{x}_4) \tag{3}$$

where $k(\boldsymbol{x}, \boldsymbol{y}) = \langle \phi(\boldsymbol{x}), \phi(\boldsymbol{y}) \rangle$ is a kernel function defined as the inner product of two objects represented in the feature space by their $\phi$ images. In the experiments reported in Section 4, we will employ a polynomial kernel, defining $k(\boldsymbol{x}, \boldsymbol{y}) = (\langle \boldsymbol{x}, \boldsymbol{y} \rangle + c)^g$, with $c = 1$ and $g = 2$. Notice that, finally we can express the ranking function $f$ in a non-linear form:

$$f(\boldsymbol{x}) = \sum_{i=1}^{n} \alpha_i z_i (k(\boldsymbol{x}_i^{(1)}, \boldsymbol{x}) - k(\boldsymbol{x}_i^{(2)}, \boldsymbol{x})) \tag{4}$$

## 3  Feature subset selection methods in a non-linear environment

When dealing with sensory data, it is important to know not only which classifier is the best and how accurate it is, but also which features are relevant for the tastes of consumers. Producers can focus on these features to improve the quality of the final product. Additionaly, reductions on the number of features often lead to a cheaper data acquisition labour, making these systems suitable for industrial operation [9].

There are many feature subset selection methods applied to SVM classification. If our goal is to find a linear separator, RFE (Recursive Feature Elimination) [10] will be a good choice. It is a ranking method that returns an ordering of the features. RFE iteratively removes the less useful feature. This process is repeated until there are no more features. Thus, we obtain an ordered sequence of features.

Following the main idea of RFE, we have used two methods capable of ordering features in non-linear scenarios. We must also point that, in this case, preference learning data sets are formed by pairs of objects $(\boldsymbol{v}, \boldsymbol{u})$, and each object in the pair has the same set of features. Thus, we must modify the ranking methods so they can deal with the duplicated features.

### 3.1  Ranking features for non-linear preference learning

**Method 1.-**  This method orders the list of features according to their influence in the variations of the weights. It is a gradient-like method, introduced in [17], and found to be a generalization of RFE to the non-linear case. It removes in each iteration the feature that minimizes the ranking value

$$R_1(i) = |\nabla_i \|\boldsymbol{w}\|^2| = \left| \sum_{k,j} \alpha_k \alpha_j z_k z_j \frac{\partial \mathcal{K}(\boldsymbol{s} \cdot \boldsymbol{x}_k, \boldsymbol{s} \cdot \boldsymbol{x}_j)}{\partial s_i} \right|, \quad i = 1, \ldots, d \tag{5}$$

where $s$ is a scaling factor used to simplify the computation of partial derivatives. Due to the fact that we are working on a preference learning problem, we need 4 copies of the scaling factor. In this formula, for a polynomial kernel $k(\boldsymbol{x}, \boldsymbol{y}) = (\langle \boldsymbol{x}, \boldsymbol{y} \rangle + c)^g$ and a vector $s$ such that $\forall i, s_i = 1$ we have that

$$\frac{\partial k(\boldsymbol{s} \cdot \boldsymbol{x}, \boldsymbol{s} \cdot \boldsymbol{y})}{\partial s_i} = 2g(x_i y_i)(c + \langle \boldsymbol{x}, \boldsymbol{y} \rangle)^{g-1}. \tag{6}$$

**Method 2.-** This method, introduced in [4], works in an iterative way; removing each time the feature which minimizes the loss of predictive performance. When using this method for preference learning with the kernel of equation (3) the ranking criterion can be expressed as

$$R_2(i) = \left( \sum_k z_k \cdot \sum_j \alpha_j z_j \mathcal{K}(\boldsymbol{x}_j^{(1),i}, \boldsymbol{x}_j^{(2),i}, \boldsymbol{x}_k^{(1),i}, \boldsymbol{x}_k^{(2),i}) \right) \tag{7}$$

where $\boldsymbol{x}^i$ denotes a vector describing an object where the value for the $i$-th feature was replaced by its mean value. Notice that a higher value of $R_2(i)$, that is, a higher accuracy on the training set when replacing feature $i$-th, means a lower relevance of that feature. Therefore, we will remove the feature yielding the highest ranking value, as opposite to the ranking method described previously.

### 3.2    Model selection on an ordered sequence of feature subsets

Once we have an ordering of the features, we must select the subset $\mathcal{F}_i$ which maximizes the generalization performance of the system. The most common choice for a model selection method is cross-validation (CV), but its efficiency and high variance [1] lead us to try another kind of methods. We have used ADJ (ADJusted distance estimate)[19]. This is a metric-based method that selects one from a nested sequence of complexity-increasing models. We construct a sequence of subsets $\mathcal{F}_1 \subset \mathcal{F}_2 \subset \ldots \subset \mathcal{F}_d$, where $\mathcal{F}_i$ represents the subset containing only the $i$ most relevant features. Then we can create a nested sequence of models $f_i$, each one of these induced by SVM from the corresponding $\mathcal{F}_i$.

The key idea is the definition of a metric on the space of hypothesis. Thus, given two different hypothesis $f$ and $g$, their distance is calculated as the expected disagreement in their predictions. Given that these distances can only be approximated, ADJ establish a method to compute $\hat{d}(g, t)$, an adjusted distance estimate between any hypothesis $f$ and the *true* target classification function $t$. Therefore, the selected hypothesis is

$$f_k = \arg\min_{f_l} \hat{d}(f_l, t). \tag{8}$$

The estimation of distance, $\hat{d}$, is computed by means of the expected disagreement in the predictions in a couple of sets: the training set $T$, and a set $U$ of unlabeled examples, that is, a set of cases sampled from the same distribution of $T$ but for which the pretended *correct* output is not given. The ADJ estimation is given by

$$ADJ(f_l, t) \stackrel{\text{def}}{=} d_T(f_l, t) \cdot \max_{k<l} \frac{d_U(f_k, f_l)}{d_T(f_k, f_l)} \tag{9}$$

where, for a given subset of examples $S$, $d_S(f, g)$ is the expected disagreement of hypothesis $f$ and $g$ in $S$. To avoid the impossibility of using the previous equation when there are zero disagreements in $T$ for two hypotheses we use the Laplace correction to the probability estimation; thus,

$$d_S(f, g) \stackrel{\text{def}}{=} \frac{1}{|S| + 2} \left( 1 + \sum_{x \in S} 1_{f(\boldsymbol{x}) \neq g(\boldsymbol{x})} \right) \tag{10}$$

In general, it is not straightforward to obtain a set of unlabeled examples. However, for learning preferences, we can easily build the set of unlabeled examples from a set of preference judgments formed by pairs of real objects randomly selected from the original preference judgment pairs.

### 3.3 Summarizing the data: dealing with redundancy

As we have previously pointed out, sensory data include ratings of experts for different characteristics of the products, as well as physical and chemical features directly measured on them. It is not infrequent to find out that some of these features are highly correlated; some experts may have similar opinions about a certain feature, and similarities among several chemical and physical features may be possible as well. In order to take advantage of these peculiarity, we have developed a simple redundancy filter, RF. It is meant to be applied before any feature subset selection method, allowing us to discover intrinsic redundancies in the data and to reduce the number of features used.

RF is an iterative process where each step gives rise to a new description of the original data set. The two most *similar* features are replaced by a new one whose values are computed as the average of them. Considering two given features $\boldsymbol{a}_i$ and $\boldsymbol{a}_j$ as (column) vectors whose dimension is the number of examples in the data set, the similarity can be estimated by means of their cosine, that is,

$$\text{similarity}(\boldsymbol{a}_i, \boldsymbol{a}_j) = \frac{\langle \boldsymbol{a}_i, \boldsymbol{a}_j \rangle}{\|\boldsymbol{a}_i\| \cdot \|\boldsymbol{a}_j\|} \tag{11}$$

Applying this method we obtain a sequence of different descriptions of the original data set, each one with one feature less than the previous. To select an adequate description in terms of prediction accuracy, we use again ADJ. The selected description can be considered a summarized version of the original data set to be processed by the feature subset selection methods previously described.

## 4 Experimental results

In this section we show the experimental results obtained when we applied the tools described in previous sections to the beef meat data base [8]. Each piece of meat was described by 147 attributes: weight of the animal, breed (7 boolean attributes), aging, 6 physical attributes describing its texture and 12 sensory traits rated by 11 different experts (132 ratings). The meat comes from 103 bovines of 7 Spanish breeds (from 13 to 16 animals of each breed); animals were slaughtered in order to obtain 54 light and 49 standard carcasses, uniformly distributed across breeds. In each rating session, 4 or 5 pieces of meat were tested and a group of consumers were asked to rate only three different aspects: tenderness, flavor and acceptance. These three data sets have over 2420 preference judgments. All the results shown in this section have been obtained by a 10-fold cross-validation.

### 4.1 Preference learning vs. regression

First, we performed a comparison between preference learning and regression methods. We have experimented with a simple linear regression and with a well reputed regression algorithm: Cubist, a commercial product from RuleQuest Research. To interpret regression results we used the relative mean absolute deviation (Rmad), which is the quotient between the mean absolute distance of the function learned by the regression method and the mean absolute distance of the constant predictor that returns the mean value in all cases.

On the other hand, we can obtain some preference judgments from the ratings of the sessions comparing the rating of each product with the rest, one by one, and constructing the

Table 1: Regression and preference learning scores on beef meat data sets.

|  | Regression | | Preferences | | | |
|---|---|---|---|---|---|---|
|  | Linear Rmad | Cubist Rmad | Linear Error | Cubist Error | $SVM_l$ Error | $SVM_p$ Error |
| tenderness | 96.3% | 97.8% | 41.5% | 43.1% | 29.6% | 19.4% |
| flavor | 99.3% | 103.4% | 43.8% | 46.5% | 32.7% | 23.8% |
| acceptance | 94.0% | 97.2% | 38.4% | 40.2% | 31.9% | 22.1% |
| Avg. | 96.5% | 99.5% | 41.2% | 43.3% | 31.4% | 21.8% |

corresponding pair. To learn from preference judgment data sets we used SVM$^{light}$ [13] with linear and polynomial kernels. In this case, the errors have a straightforward meaning as misclassifications; so in order to allow a fair comparison between regression and preference learning approaches, we also tested regression models on preference judgments test sets, calculating their misclassifications. The scores achieved on the three data sets described previously, are shown in Table 1.

We can observe that regression methods are unable to learn any useful knowledge: their relative mean absolute deviation (Rmad) is near 100% in all cases, that is, regression models usually perform equal than the constant predictor forecasting the mean value. From a practical point of view, these results mean that raw consumers' ratings can not be used to measure the overall sensory opinions. Even when these regression models are tested on preference judgment sets the percentage of misclassifications is over 40%, clearly higher than those obtained when using the preference learning approach. SVM-based methods can reduce these errors down to an average near 30% with a linear kernel ($SVM_l$), and near 20% if the kernel is a polynomial of degree 2 ($SVM_p$). This improvement shows that non-linear kernels can explain consumers preferences better.

## 4.2 Feature selection

We used the FSS tools to gain insight into consumer preferences. For the sake of simplicity, in what follows $FSS_1$ and $FSS_2$ will denote the feature subset selectors that use ranking *Method 1* and *Method 2* respectively. The learner used in these experiments was $SVM_p$ because it was the most accurate in previous tests (see Table 1). Given the size of the three data sets it is almost impractical to use $FSS_1$ and $FSS_2$ due to its computational cost, unless a previous reduction in the number of features can be achieved; therefore, in both cases we used RF as a previous filter. Additionally, to improve the overall speed, features were removed in chunks of five. In all cases we used ADJ to choose among the subsets of features. We can see in Table 2 that $FSS_1$, $FSS_2$, and RF considerably reduce the number of features at the expense of accuracy: it slightly decreases when we use the RF filter with respect to the accuracy obtained on the original data set by $SVM_p$; it also decreases when using $FSS_1$ and $FSS_2$ after RF.

The most useful result obtained from feature selection is the ranking list of traits. We concentrate our study in tenderness and acceptance categories because they are more interesting from the point of view of beef meat producers. So, in acceptance data set, the three most useful traits are: aging, breed and fibrosis. Some research works in the beef meat field corroborate the importance of these characteristics [18, 12]. Specially, aging is crucial to improve consumer acceptance. On the other hand, fibrosis is closely related with tenderness: the less fibrosis, the more tenderness. Usually many consumers identify tenderness with acceptance, in the sense that a higher tenderness yields to a higher acceptance; then, fibrosis and acceptance are inversely related. With respect to the breed trait, two of the

Table 2: Percentage of misclassifications and the number of selected features when polynomial kernel ($SVM_p$) and FSS methods are used. The three original data sets have 147 features.

|  | RF | | RF+FSS$_1$ | | RF+FSS$_2$ | |
|---|---|---|---|---|---|---|
|  | Error | #Att. | Error | #Att. | Error | #Att. |
| tenderness | 20.0% | 50.0 | 21.8% | 27.0 | 21.3% | 37.5 |
| flavor | 25.0% | 65.0 | 26.5% | 33.5 | 26.1% | 29.0 |
| acceptance | 24.7% | 39.5 | 24.8% | 30.0 | 25.3% | 26.7 |
| Avg. | 23.2% | 51.5 | 24.4% | 30.2 | 24.2% | 31.1 |

seven possible values, *retinta* and *asturiana* breeds [8], have more influence than others in the preference function that describes consumer acceptance; for example, meat from *retinta* animals seems to be the most appreciated by consumers.

In tenderness data set, the most useful attributes are: aging, fibrosis, residue and odor intensity. Aging and fibrosis appear again, showing the relationship between acceptance and tenderness. Residue depends on fibrosis, so it is not a surprise to find it in the list. Apparently, odor intensity is not so related to tenderness, but it is closely related to aging.

## 5    Conclusions

We have shown that an approach based on nonlinear SVMs can be useful to model consumer preferences about beef meat. The polynomial model obtained and the FSS tools used allow us to emphasize the relevance of meat traits previously described in the literature of the field. However, the novelty of our approach is that we can algorithmically deduce the expressions of relevance.

The sensory data base available probably tries to cover too many aspects of beef meat affecting to its sensory quality. Therefore, it is not possible to obtain more detailed conclusions from the polynomial model. Nevertheless, the experience reported in this paper can be very useful for the design of future experiments involving specific traits of beef meat quality.

### Acknowledgements

The research reported in this paper has been supported in part under Spanish Ministerio de Ciencia y Tecnología (MCyT) and Feder grant TIC2001-3579.

## References

[1]  A. Bahamonde, G. F. Bayón, J. Díez, J. R. Quevedo, O. Luaces, J. J. del Coz, J. Alonso, and F. Goyache. Feature subset selection for learning preferences: A case study. In *Proceedings of the International Conference on Machine Learning*, Banff, Alberta (Canada), July 2004. Morgan Kaufmann.

[2]  D. Buck, I. Wakeling, K. Greenhoff, and A. Hasted. Predicting paired preferences from sensory data. *Food quality and preference*, 12:481–487, 2001.

[3]  D. Corney. Designing food with bayesian belief networks. In *Proceedings of the International Conference on Adaptive Computing in engineering Design and Manufacture*, pages 83–94, 2002.

[4]  S. Degroeve, B. De Baets, Y. Van de Peer, and P. Rouzé. Feature subset selection for splice site prediction. *Bioinformatics*, 18(2):75–83, 2002.

[5] J. Díez, A. Bahamonde, J. Alonso, S. López, J. del Coz, J. Quevedo, J. Ranilla, O. Luaces, I. Álvarez, L. Royo, and F. Goyache. Artificial intelligence techniques point out differences in classification performance between light and standard bovine carcasses. *Meat Science*, 64(3):249–258, 2003.

[6] J. Díez, G. F. Bayón, J. R. Quevedo, J. J. del Coz, O. Luaces, J. Alonso, and A. Bahamonde. Discovering relevancies in very difficult regression problems: applications to sensory data analysis. In *Proceedings of the European Conference on Artificial Intelligence (ECAI '04)*, Valencia, Spain, 2004.

[7] S. Dumais, K. Bharat, T. Joachims, and A. Weigend, editors. *Workshop on implicit measures of user interests and preferences. In ACM SIGIR Conference*, Toronto, Canada, 2003.

[8] M. Gil, X. Serra, M. Gispert, M. Oliver, C. Sañudo, B. Panea, J. Olleta, M. Campo, M. Oliván, K. Osoro, M. Garcia-Cachan, R. Cruz-Sagredo, M. Izquierdo, M. Espejo, M. Martín, and J. Piedrafita. The effect of breed-production systems on the myosin heavy chain 1, the biochemical characteristics and the colour variables of longissimus thoracis from seven spanish beef cattle breeds. *Meat Science*, 58:181–188, 2001.

[9] F. Goyache, A. Bahamonde, J. Alonso, S. López, del Coz J.J., J. Quevedo, J. Ranilla, O. Luaces, I. Alvarez, L. Royo, and J. Díez. The usefulness of artificial intelligence techniques to assess subjective quality of products in the food industry. *Trends in Food Science and Technology*, 12(10):370–381, 2001.

[10] I. Guyon, J. Weston, S. Barnhill, and V. Vapnik. Gene selection for cancer classification using support vector machines. *Machine Learning*, 46(1–3):389–422, 2002.

[11] R. Herbrich, T. Graepel, and K. Obermayer. Large margin rank boundaries for ordinal regression. In A. Smola, P. Bartlett, B. Scholkopf, and D. Schuurmans, editors, *Advances in Large Margin Classifiers*, pages 115–132. MIT Press, Cambridge, MA, 2000.

[12] L. Jeremiah and L. Gibson. The effects of postmortem product handling and aging time on beef palatability. *Food Research International*, 36:929–941, 2003.

[13] T. Joachims. Making large-scale support vector machines learning practical. In A. S. B. Schölkopf, C. Burges, editor, *Advances in Kernel Methods: Support Vector Machines*. MIT Press, Cambridge, MA, 1998.

[14] T. Joachims. Optimizing search engines using clickthrough data. In *Proceedings of the ACM Conference on Knowledge Discovery and Data Mining (KDD)*, 2002.

[15] J. Murray, C. Delahunty, and I. Baxter. Descriptive sensory analysis: past, present and future. *Food Research International*, 36:461–471, 2001.

[16] T. Næs and E. Risvik. *Multivariate analysis of data in sensory science*. Elsevier, 1996.

[17] A. Rakotomamonjy. Variable selection using SVM-based criteria. *Journal of Machine Learning Research*, 3:1357–1370, 2003.

[18] C. Sañudo, E. S. Macie, J. L. Olleta, M. Villaroel, B. Panea, and P. Albertí. The effects of slaughter weight, breed type and ageing time on beef meat quality using two different texture devices. *Meat Science*, 66:925–932, 2004.

[19] D. Schuurmans and F. Southey. Metric-based methods for adaptive model selection and regularization. *Machine Learning*, 48:51–84, 2002.
